# Using a neural net to instantiate a deformable model

Christopher K. I. Williams,* Michael D. Revow and Geoffrey E. Hinton
Department of Computer Science, University of Toronto
Toronto, Ontario, Canada M5S 1A4

## Abstract

Deformable models are an attractive approach to recognizing non-rigid objects which have considerable within class variability. However, there are severe search problems associated with fitting the models to data. We show that by using neural networks to provide better starting points, the search time can be significantly reduced. The method is demonstrated on a character recognition task.

In previous work we have developed an approach to handwritten character recognition based on the use of deformable models (Hinton, Williams and Revow, 1992a; Revow, Williams and Hinton, 1993). We have obtained good performance with this method, but a major problem is that the search procedure for fitting each model to an image is very computationally intensive, because there is no efficient algorithm (like dynamic programming) for this task. In this paper we demonstrate that it is possible to "compile down" some of the knowledge gained while fitting models to data to obtain better starting points that significantly reduce the search time.

## 1 DEFORMABLE MODELS FOR DIGIT RECOGNITION

The basic idea in using deformable models for digit recognition is that each digit has a model, and a test image is classified by finding the model which is most likely to have generated it. The quality of the match between model and test image depends on the deformation of the model, the amount of ink that is attributed to noise and the distance of the remaining ink from the deformed model.

More formally, the two important terms in assessing the fit are the prior probability distribution for the instantiation parameters of a model (which penalizes very distorted models), and the imaging model that characterizes the probability distribution over possible images given the instantiated model[1]. Let $I$ be an image, $M$ be a model and $z$ be its instantiation parameters. Then the evidence for model $M$ is given by

$$P(I|M) = \int P(z|M)P(I|M,z)dz \qquad (1)$$

The first term in the integrand is the prior on the instantiation parameters and the second is the imaging model i.e., the likelihood of the data given the instantiated model. $P(M|I)$ is directly proportional to $P(I|M)$, as we assume a uniform prior on each digit.

Equation 1 is formally correct, but if $z$ has more than a few dimensions the evaluation of this integral is very computationally intensive. However, it is often possible to make an approximation based on the assumption that the integrand is strongly peaked around a (global) maximum value $z^*$. In this case, the evidence can be approximated by the highest peak of the integrand times a volume factor $\Delta(z|I,M)$, which measures the sharpness of the peak[2].

$$P(I|M) \simeq P(z^*|M)P(I|z^*,M)\Delta(z|I,M) \qquad (2)$$

By Taylor expanding around $z^*$ to second order it can be shown that the volume factor depends on the determinant of the Hessian of $\log P(z,I|M)$ . Taking logs of equation 2, defining $E_{def}$ as the negative log of $P(z^*|M)$, and $E_{fit}$ as the corresponding term for the imaging model, then the aim of the search is to find the minimum of $E_{tot} = E_{def} + E_{fit}$. Of course the total energy will have many local minima; for the character recognition task we aim to find the global minimum by using a continuation method (see section 1.2).

## 1.1   SPLINES, AFFINE TRANSFORMS AND IMAGING MODELS

This section presents a brief overview of our work on using deformable models for digit recognition. For a fuller treatment, see Revow, Williams and Hinton (1993).

Each digit is modelled by a cubic B-spline whose shape is determined by the positions of the control points in the object-based frame. The models have eight control points, except for the one model which has three, and the seven model which has five. To generate an ideal example of a digit the control points are positioned at their "home" locations. Deformed characters are produced by perturbing the control points away from their home locations. The home locations and covariance matrix for each model were adapted in order to improve the performance.

The deformation energy only penalizes shape *deformations*. Affine transformations, i.e., translation, rotation, dilation, elongation, and shear, do not change the underlying shape of an object so we want the deformation energy to be invariant under them. We achieve this by giving each model its own "object-based frame" and computing the deformation energy relative to this frame.

The data we used consists of binary-pixel images of segmented handwritten digits. The general flavour of a imaging model for this problem is that there should be a high probability of inked pixels close to the spline, and lower probabilities further away. This can be achieved by spacing out a number of Gaussian "ink generators" uniformly along the contour; we have found that it is also useful to have a uniform background noise process over the area of the image that is able to account for pixels that occur far away from the generators. The ink generators and background process define a mixture model. Using the assumption that each data point is generated independently given the instantiated model, $P(I|z^*, M)$ factors into the product of the probability density of each black pixel under the mixture model.

## 1.2 RECOGNIZING ISOLATED DIGITS

For each model, the aim of the search is to find the instantiation parameters that minimize $E_{tot}$. The search starts with zero deformations and an initial guess for the affine parameters which scales the model so as to lie over the data with zero skew and rotation. A small number of generators with the same large variance are placed along the spline, forming a broad, smooth ridge of high ink-probability along the spline. We use a search procedure similar to the (iterative) Expectation Maximization (EM) method of fitting an unconstrained mixture of Gaussians, except that (i) the Gaussians are constrained to lie on the spline (ii) there is a deformation energy term and (iii) the affine transformation must be recalculated on each iteration. During the search the number of generators is gradually increased while their variance decreases according to predetermined "annealing" schedule[3].

After fitting all the models to a particular image, we wish to evaluate which of the models best "explains" the data. The natural measure is the sum of $E_{fit}$, $E_{def}$ and the volume factor. However, we have found that performance is improved by including four additional terms which are easily obtained from the final fits of the model to the image. These are (i) a measure which penalizes matches in which there are beads far from any inked pixels (the "beads in white space" problem), and (ii) the rotation, shear and elongation of the affine transform. It is hard to decide in a principled way on the correct weightings for all of these terms in the evaluation function. We estimated the weightings from the data by training a simple postprocessing neural network. These inputs are connected directly to the ten output units. The output units compete using the "softmax" function which guarantees that they form a probability distribution, summing to one.

## 2 PREDICTING THE INSTANTIATION PARAMETERS

The search procedure described above is very time consuming. However, given many examples of images and the corresponding instantiation parameters obtained by the slow method, it is possible to train a neural network to predict the instantiation parameters of novel images. These predictions provide better starting points, so the search time can be reduced.

## 2.1  PREVIOUS WORK

Previous work on hypothesizing instantiation parameters can be placed into two broad classes, correspondence based search and parameter space search. In correspondence based search, the idea is to extract features from the image and identify corresponding features in the model. Using sufficient correspondences the instantiation parameters of the model can be determined. The problem is that simple, easily detectable image features have many possible matches, and more complex features require more computation and are more difficult to detect. Grimson (1990) shows how to search the space of possible correspondences using an interpretation tree.

An alternative approach, which is used in Hough transform techniques, is to directly work in parameter space. The Hough transform was originally designed for the detection of straight lines in images, and has been extended to cover a number of geometric shapes, notably conic sections. Ballard (1981) further extended the approach to arbitrary shapes with the Generalized Hough Transform. The parameter space for each model is divided into cells ("binned"), and then for each image feature a vote is added to each parameter space bin that could have produced that feature. After collecting votes from all image features we then search for peaks in the parameter space accumulator array, and attempt to verify pose. The Hough transform can be viewed as a crude way of approximating the logarithm of the posterior distribution $P(z|I, M)$ (e.g. Hunt *et al*, 1988).

However, these two techniques have only been used on problems involving rigid models, and are not readily applicable to the digit recognition problem. For the Hough space method, binning and vote collection is impractical in the high dimensional parameter space, and for the correspondence based approach there is a lack of easily identified and highly discriminative features. The strengths of these two techniques, namely their ability to deal with arbitrary scalings, rotations and translations of the data, and their tolerance of extraneous features, are not really required for a task where the input data is fairly well segmented and normalized.

Our approach is to use a neural network to predict the instantiation parameters for each model, given an input image. Zemel and Hinton (1991) used a similar method with simple 2-d objects, and more recently, Beymer *et al* (1993) have constructed a network which maps from a face image to a 2-d parameter space spanning head rotations and a smile/no-smile dimension. However, their method does not directly map from images to instantiation parameters; they use a computer vision correspondence algorithm to determine the displacement field of pixels in a novel image relative to a reference image, and then use this field as the input to the network. This step limits the use of the approach to images that are sufficiently similar so that the correspondence algorithm functions well.

## 2.2  INSTANTIATING DIGIT MODELS USING NEURAL NETWORKS

The network which is used to predict the model instantiation parameters is shown in figure 1. The (unthinned) binary images are normalized to give $16 \times 16$ 8-bit greyscale images which are fed into the neural network. The network uses a standard three-layer architecture; each hidden unit computes a weighted sum of its inputs, and then feeds this value through a sigmoidal nonlinearity $\sigma(x) = 1/(1 + e^{-x})$. The

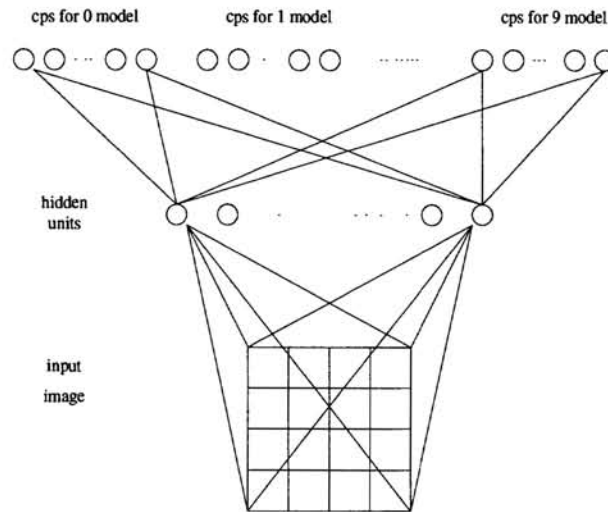

Figure 1: The prediction network architecture. "cps" stands for control points.

output values are a weighted linear combination of the hidden unit activities plus output biases. The targets are the locations of the control points in the normalized image, found from fitting models as described in section 1.2.

The network was trained with backpropagation to minimize the squared error, using 900 training images and 200 validation images of each digit drawn from the *br* set of the CEDAR CDROM 1 database of Cities, States, ZIP Codes, Digits, and Alphabetic Characters[4]. Two test sets were used; one was obtained from data in the *br* dataset, and the other was the (official) *bs* test set. After some experimentation we chose a network with twenty hidden units, which means that the net has over 8,000 weights. With such a large number of weights it is important to regularize the solution obtained by the network by using a complexity penalty; we used a weight penalty $\lambda \sum_j w_j^2$ and optimized $\lambda$ on a validation set. Targets were only set for the correct digit at the output layer; nothing was backpropagated from the other output units. The net took 440 epochs to train using the default conjugate gradient search method in the Xerion neural network simulator[5]. It would be possible to construct ten separate networks to carry out the same task as the net described above, but this would intensify the danger of overfitting, which is reduced by giving the network a common pool of hidden units which it can use as it decides appropriate.

For comparison with the *prediction net* described above, a trivial network which just consisted of output biases was trained; this network simply learns the average value of the control point locations. On a validation set the squared error of the prediction net was over three times smaller than the trivial net. Although this is encouraging, the acid test is to compare the performance of elastic models settled from the predicted positions using a shortened annealing schedule; if the predictions are good, then only a short amount of settling will be required.

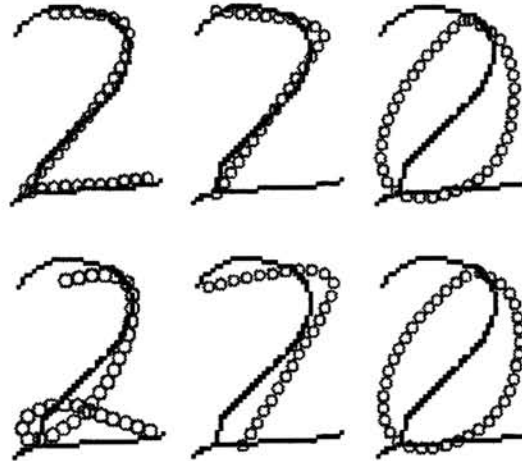

Figure 2: A comparision of the initial instantiations due to the prediction net (top row) and the trivial net (bottom row) on an image of a 2. Notice that for the two model the prediction net is much closer to the data. The other digit models may or may not be greatly affected by the input data; for example, the predictions from both nets seem essentially the same for the zero, but for the seven the prediction net puts the model nearer to the data.

The feedforward net predicts the position of the control points in the normalized image. By inverting the normalization process, the positions of the control points in the un-normalized image are determined. The model deformation and affine transformation corresponding to these image control point locations can then be determined by running a part of one iteration of the search procedure. Experiments were then conducted with a number of shortened annealing schedules; for each one, data obtained from settling on a part of the training data was used to train the postprocessing net. The performance was then evaluated on the *br* test set.

The full annealing schedule has six stages. The shortened annealing schedules are:

1. No settling at all
2. Two iterations at the final variance of 0.0006
3. One iteration at 0.0025 and two at 0.0006
4. The full annealing schedule (for comparison)

The results on the *br* test set are shown in table 1. The general trends are that the performance obtained using the prediction net is consistently better than the trivial net, and that longer annealing schedules lead to better performance. A comparison of schedules 3 and 4 in table 1 indicates that the performance of the prediction net/schedule 3 combination is similar to (or slightly better than) that obtained with the full annealing schedule, and is more than a factor of two faster. The results with the full schedule are almost identical to the results obtained with the default "box" initialization described in section 1.2. Figure 2 compares the outputs of the prediction and trivial nets on a particular example. Judging from the weight

| Schedule number | Trivial net | Prediction net | Average time required to settle one model (s) |
|:---:|:---:|:---:|:---:|
| 1 | 427 | 200 | 0.12 |
| 2 | 329 | 58 | 0.25 |
| 3 | 160 | 32 | 0.49 |
| 4 | 40 | 36 | 1.11 |

Table 1: Errors on the internal test set of 2000 examples for different annealing schedules. The timing trials were carried out on a R-4400 machine.

vectors and activity patterns of the hidden units, it does not seem that some of the units are specialized for a particular digit class.

A run on the *bs* test set using schedule 3 gave an error rate of 4.76 % (129 errors), which is very similar to the 125 errors obtained using the full annealing schedule and the box initialization. A comparison of the errors made on the two runs shows that only 67 out of the 129 errors were common to the two sets. This suggests that it would be very sensible to reject cases where the two methods do not agree.

# 3   DISCUSSION

The prediction net used above can be viewed as an interpolation scheme in the control point position space of each digit $z(I) = z_0 + \sum_i a_i(I)z_i$, where $z(I)$ is the predicted position in the control point space, $z_0$ is the contribution due to the biases, $a_i$ is the activity of hidden unit $i$ and $z_i$ is its location in the control point position space (learned from the data). If there are more hidden units than output dimensions, then for any particular image there are an infinite number of ways to make this equation hold exactly. However, the network will tend to find solutions so that the $a_i(I)$'s will vary smoothly as the image is perturbed.

The nets described above output just one set of instantiation parameters for a given model. However, it may be preferable to be able to represent a number of guesses about model instantiation parameters; one way of doing this is to train a network that has multiple sets of output parameters, as in the "mixture of experts" architecture of Jacobs *et al* (1991). The outputs can be interpreted as a mixture distribution in the control point position space, conditioned on the input image. Another approach to providing more information about the posterior distribution is described in (Hinton, Williams and Revow, 1992b), where $P(z|I)$ is approximated using a fixed set of basis functions whose weighting depends on the input image $I$.

The strategies described above directly predict the instantiation parameters in parameter space. It is also possible to use neural networks to hypothesize correspondences, i.e. to predict an inked pixel's position on the spline given a local window of context in the image. With sufficient matches it is then possible to compute the instantiation parameters of the model. We have conducted some preliminary experiments with this method (described in Williams, 1994), which indicate that good performance can be achieved for the correspondence prediction task.

We have shown that the we can obtain significant speedup using the prediction net. The schemes outlined above which allow multimodal predictions in instantiation parameter space may improve performance and deserve further investigation. We are also interested in improving the performance of the prediction net, for example by outputting a confidence measure which could be used to adjust the length of the elastic models' search appropriately. We believe that using machine learning techniques like neural networks to help reduce the amount of search required to fit complex models to data may be useful for many other problems.

## Acknowledgements

This research was funded by Apple and by the Ontario Information Technology Research Centre. We thank Allan Jepson, Richard Durbin, Rich Zemel, Peter Dayan, Rob Tibshirani and Yann Le Cun for helpful discussions. Geoffrey Hinton is the Noranda Fellow of the Canadian Institute for Advanced Research.

## Footnotes

*Current address: Department of Computer Science and Applied Mathematics, Aston University, Birmingham B4 7ET, UK.

[1]This framework has been used by many authors, e.g. Grenander *et al* (1991).

[2]The Gaussian approximation has been popularized in the neural net community by MacKay (1992).

[3]The schedule starts with 8 beads increasing to 60 beads in six steps, with the variance decreasing from 0.04 to 0.0006 (measured in the object frame). The scale is set in the object-based frame so that each model is 1 unit high.

[4]Made available by the Unites States Postal Service Office of Advanced Technology.

[5]Xerion was designed and implemented by Drew van Camp, Tony Plate and Geoffrey Hinton at the University of Toronto.

## References

Ballard, D. H. (1981). Generalizing the Hough transfrom to detect arbitrary shapes. *Pattern Recognition*, 13(2):111–122.

Beymer, D., Shashua, A., and Poggio, T. (1993). Example Based Image Analysis and Synthesis. AI Memo 1431, AI Laboratory, MIT.

Grenander, U., Chow, Y., and Keenan, D. M. (1991). *Hands: A pattern theoretic study of biological shapes.* Springer-Verlag.

Grimson, W. E. L. (1990). *Object recognition by computer.* MIT Press, Cambridge, MA.

Hinton, G. E., Williams, C. K. I., and Revow, M. D. (1992a). Adaptive elastic models for hand-printed character recognition. In Moody, J. E., Hanson, S. J., and Lippmann, R. P., editors, *Advances in Neural Information Processing Systems 4.* Morgan Kauffmann.

Hinton, G. E., Williams, C. K. I., and Revow, M. D. (1992b). Combinining two methods of recognizing hand-printed digits. In Aleksander, I. and Taylor, J., editors, *Artificial Neural Networks 2.* Elsevier Science Publishers.

Hunt, D. J., Nolte, L. W., and Ruedger, W. H. (1988). Performance of the Hough Transform and its Relationship to Statistical Signal Detection Theory. *Computer Vision, Graphics and Image Processing*, 43:221–238.

Jacobs, R. A., Jordan, M. I., Nowlan, S. J., and Hinton, G. E. (1991). Adaptive mixtures of local experts. *Neural Computation*, 3(1).

MacKay, D. J. C. (1992). Bayesian Interpolation. *Neural Computation*, 4(3):415–447.

Revow, M. D., Williams, C. K. I., and Hinton, G. E. (1993). Using mixtures of deformable models to capture variations in hand printed digits. In Srihari, S., editor, *Proceedings of the Third International Workshop on Frontiers in Handwriting Recognition*, pages 142–152, Buffalo, New York, USA.

Williams, C. K. I. (1994). *Combining deformable models and neural networks for hand-printed digit recognition.* PhD thesis, Dept. of Computer Science, University of Toronto.

Zemel, R. S. and Hinton, G. E. (1991). Discovering viewpoint-invariant relationships that characterize objects. In Lippmann, R. P., Moody, J. E., and Touretzky, D. S., editors, *Advances In Neural Information Processing Systems 3*, pages 299–305. Morgan Kaufmann Publishers.
